# Functional Geometry Alignment and Localization of Brain Areas

**Georg Langs, Polina Golland**
Computer Science and Artificial Intelligence Lab, Massachusetts Institute of Technology
Cambridge, MA 02139, USA
langs@csail.mit.edu, polina@csail.mit.edu

**Yanmei Tie, Laura Rigolo, Alexandra J. Golby**
Department of Neurosurgery, Brigham and Women's Hospital, Harvard Medical School
Boston, MA 02115, USA
ytie@bwh.harvard.edu, lrigolo@bwh.harvard.edu
agolby@bwh.harvard.edu

## Abstract

Matching functional brain regions across individuals is a challenging task, largely due to the variability in their location and extent. It is particularly difficult, but highly relevant, for patients with pathologies such as brain tumors, which can cause substantial reorganization of functional systems. In such cases spatial registration based on anatomical data is only of limited value if the goal is to establish correspondences of functional areas among different individuals, or to localize potentially displaced active regions. Rather than rely on spatial alignment, we propose to perform registration in an alternative space whose geometry is governed by the functional interaction patterns in the brain. We first embed each brain into a functional map that reflects connectivity patterns during a fMRI experiment. The resulting functional maps are then registered, and the obtained correspondences are propagated back to the two brains. In application to a language fMRI experiment, our preliminary results suggest that the proposed method yields improved functional correspondences across subjects. This advantage is pronounced for subjects with tumors that affect the language areas and thus cause spatial reorganization of the functional regions.

## 1 Introduction

Alignment of functional neuroanatomy across individuals forms the basis for the study of the functional organization of the brain. It is important for localization of specific functional regions and characterization of functional systems in a population. Furthermore, the characterization of variability in location of specific functional areas is itself informative of the mechanisms of brain formation and reorganization. In this paper we propose to align neuroanatomy based on the functional geometry of fMRI signals during specific cognitive processes. For each subject, we construct a map based on spectral embedding of the functional connectivity of the fMRI signals and register those maps to establish correspondence between functional areas in different subjects.

Standard registration methods that match brain anatomy, such as the Talairach normalization [21] or non-rigid registration techniques like [10, 20], accurately match the anatomical structures across individuals. However the variability of the functional locations relative to anatomy can be substantial [8, 22, 23], which limits the usefulness of such alignment in functional experiments. The relationship between anatomy and function becomes even less consistent in the presence of pathological changes in the brain, caused by brain tumors, epilepsy or other diseases [2, 3].

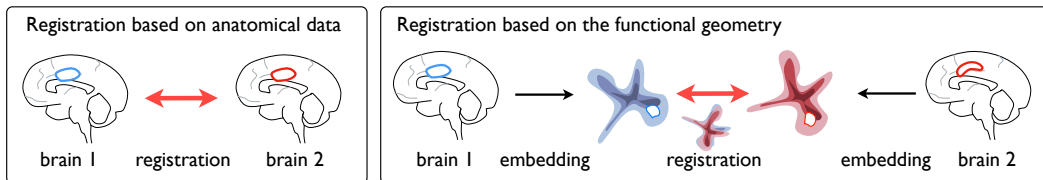

Figure 1: Standard anatomical registration and the proposed functional geometry alignment. Functional geometry alignment matches the diffusion maps of fMRI signals of two subjects.

Integrating functional features into the registration process promises to alleviate this challenge. Recently proposed methods match the centers of activated cortical areas [22, 26], or estimate dense correspondences of cortical surfaces [18]. The fMRI signals at the surface points serve as a feature vector, and registration is performed by maximizing the inter-subject fMRI correlation of matched points, while at the same time regularizing the surface warp to preserve cortical topology and penalizing cortex folding and metric distortion, similar to [9]. In [7] registration of a population of subjects is accomplished by using the functional connectivity pattern of cortical points as a descriptor of the cortical surface. It is warped so that the Frobenius norm of the difference between the connectivity matrices of the reference subject and the matched subject is minimized, while at the same time a topology-preserving deformation of the cortex is enforced.

All the methods described above rely on a spatial reference frame for registration, and use the functional characteristics as a feature vector of individual cortical surface points or the entire surface. This might have limitations in the case of severe pathological changes that cause a substantial reorganization of the functional structures. Examples include the migration to the other hemisphere, changes in topology of the functional maps, or substitution of functional roles played by one damaged region with another area. In contrast, our approach to functional registration does not rely on spatial consistency.

Spectral embedding [27] represents data points in a map that reflects a large set of measured pairwise affinity values in a euclidean space. Previously we used spectral methods to map voxels of a fMRI sequence into a space that captures joint functional characteristics of brain regions [14]. This approach represents the level of interaction by the density in the embedding. In [24], different embedding methods were compared in a study of parceled resting-state fMRI data. Functionally homogeneous units formed clusters in the embedding. In [11] multidimensional scaling was employed to retrieve a low dimensional representation of positron emission tomography (PET) signals after selecting sets of voxels by the standard activation detection technique [12].

Here we propose and demonstrate a functional registration method that operates in a space that reflects functional connectivity patterns of the brain. In this space, the connectivity structure is captured by a structured distribution of points, or *functional geometry*. Each point in the distribution represents a location in the brain and the relation of its fMRI signal to signals at other locations. Fig. 1 illustrates the method. To register functional regions among two individuals, we first embed both fMRI volumes independently, and then obtain correspondences by matching the two point distributions in the functional geometry. We argue that such a representation offers a more natural view of the co-activation patterns than the spatial structure augmented with functional feature vectors. The functional geometry can handle long-range reorganizations and topological variability in the functional organization of different individuals. Furthermore, by translating connectivity strength to distances we are able to regularize the registration effectively. Strong connections are preserved during registration in the map by penalizing high-frequencies in the map deformation field.

The clinical goal of our work is to reliably localize language areas in tumor patients. The functional connectivity pattern for a specific area provides a refined representation of its activity that can augment the individual activation pattern. Our approach is to utilize connectivity information to improve localization of the functional areas in tumor patients. Our method transfers the connectivity patterns from healthy subjects to tumor patients. The transferred patterns then serve as a patient-specific prior for functional localization, improving the accuracy of detection. The functional geometry we use is largely independent of the underlying anatomical organization. As a consequence, our method handles substantial changes in spatial arrangement of the functional areas that typically present sig-

nificant challenges for anatomical registration methods. Such functional priors promise to improve detection accuracy in the challenging case of language area localization. The mapping of healthy networks to patients provides additional evidence for the location of the language areas. It promises to enhance accuracy and robustness of localization.

In addition to localization, studies of reconfiguration mechanisms in the presence of lesions aim to understand how specific sub-areas are redistributed (e.g., do they migrate to a compact area, or to other intact language areas). While standard detection identifies the regions whose activation is correlated with the experimental protocol, we seek a more detailed description of the functional roles of the detected regions, based on the functional connectivity patterns.

We evaluate the method on healthy control subjects and brain tumor patients who perform language mapping tasks. The language system is highly distributed across the cortex. Tumor growth sometimes causes a reorganization that sustains language ability of the patient, even though the anatomy is severely changed. Our initial experimental results indicate that the proposed functional alignment outperforms anatomical registration in predicting activation in target subjects (both healthy controls and patients). Furthermore functional alignment can handle substantial reorganizations and is much less affected by the tumor presence than anatomical registration.

## 2 Embedding the brain in a functional geometry

We first review the representation of the functional geometry that captures the co-activation patterns in a diffusion map defined on the fMRI voxels [6, 14]. Given a fMRI sequence $\mathbf{I} \in \mathbb{R}^{T \times N}$ that contains $N$ voxels, each characterized by an fMRI signal over $T$ time points, we calculate matrix $\mathbf{C} \in \mathbb{R}^{N \times N}$ that assigns each pair of voxels $\langle k, l \rangle$ with corresponding time courses $\mathbf{I}_k$ and $\mathbf{I}_l$ a non-negative symmetric weight

$$c(k, l) = e^{\frac{corr(\mathbf{I}_k, \mathbf{I}_l)}{\epsilon}}, \tag{1}$$

where $corr$ is the correlation coefficient of the two signals $\mathbf{I}_k$ and $\mathbf{I}_l$, and $\epsilon$ is the speed of weight decay. We define a graph whose vertices correspond to voxels and whose edge weights are determined by $\mathbf{C}$. In practice, we discard all edges that have a weight below a chosen threshold if they connect nodes with a large distance in the anatomical space. This construction yields a sparse graph which is then transformed into a Markov chain. Note that in contrast to methods like multidimensional scaling, this sparsity reflects the intuition that meaningful information about the connectivity structure is encoded by the high correlation values.

We transform the graph into a Markov chain on the set of nodes by the normalized graph Laplacian construction [5]. The degree of each node $g(k) = \sum_l c(k, l)$ is used to define the directed edge weights of the Markov chain as

$$p(k, l) = \frac{c(k, l)}{g(k)}, \tag{2}$$

which can be interpreted as transition probabilities along the graph edges. This set of probabilities defines a diffusion operator $Pf(x) = \sum p(x, y)f(y)$ on the graph vertices (voxels). The diffusion operator integrates all pairwise relations in the graph and defines a geometry on the entire set of fMRI signals.

We embed the graph in a Euclidean space via an eigenvalue decomposition of $P$ [6]. The eigenvalue decomposition of the operator $P$ results in a sequence of decreasing eigen values $\lambda_1, \lambda_2 \ldots$ and corresponding eigen vectors $\Psi_1, \Psi_2, \ldots$ that satisfy $P\Psi_i = \lambda_i \Psi_i$ and constitute the so-called *diffusion map*:

$$\mathbf{\Psi}_t \triangleq \langle \lambda_1^t \Psi_1 \ldots \lambda_w^t \Psi_w \rangle, \tag{3}$$

where $w \leq T$ is the dimensionality of the representation, and $t$ is a parameter that controls scaling of the axes in this newly defined space. $\mathbf{\Psi}_t^k \in \mathbb{R}^w$ is the representation of voxel $k$ in the functional geometry; it comprises the $k$th components of the first $w$ eigenvectors. We will refer to $\mathbb{R}^w$ as the *functional space*. The global structure of the functional connectivity is reflected in the point distribution $\mathbf{\Psi}_t$. The axes of the eigenspace are the directions that capture the highest amount of structure in the connectivity landscape of the graph.

This functional geometry is governed by the diffusion distance $D_t$ on the graph: $D_t(k, l)$ is defined through the probability of traveling between two vertices $k$ and $l$ by taking all paths of at most $t$ steps

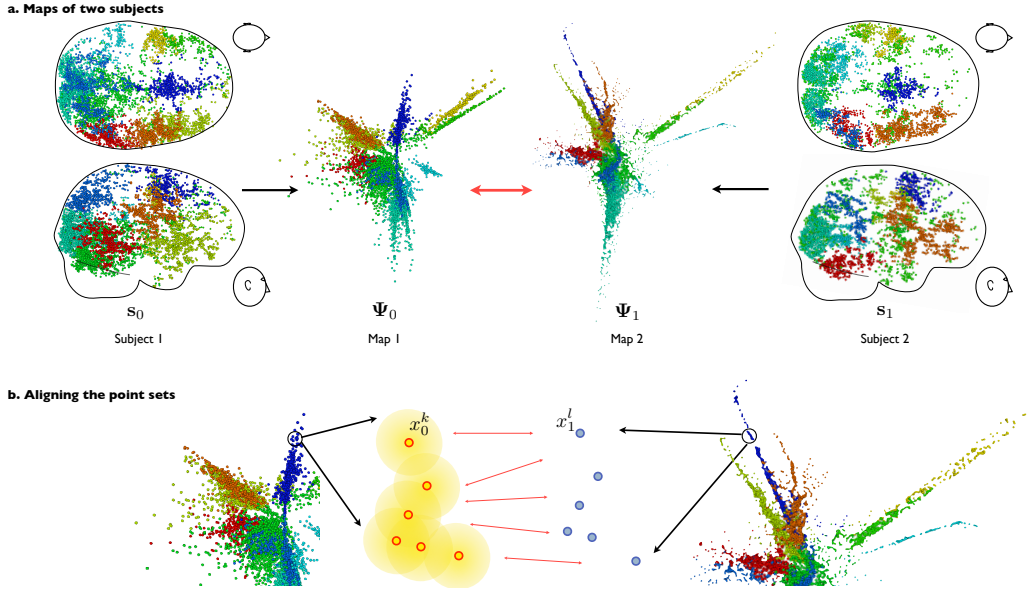

**a. Maps of two subjects**

$\mathbf{s}_0$    $\boldsymbol{\Psi}_0$    $\boldsymbol{\Psi}_1$    $\mathbf{s}_1$

Subject I    Map I    Map 2    Subject 2

**b. Aligning the point sets**

$x_0^k$    $x_1^l$

Figure 2: Maps of two subjects in the process of registration: (a) Left and right: the axial and sagittal views of the points in the two brains. The two central columns show plots of the first three dimensions of the embedding in the functional geometry after coarse rotational alignment. (b) During alignment, a maps is represented as a Gaussian mixture model. The colors in both plots indicate clusters which are only used for visualization.

into account. The transition probabilities are based on the functional connectivity of pairs of nodes. Thus the diffusion distance integrates the connectivity values over possible paths that connect two points and defines a geometry that captures the entirety of the connectivity structure. It corresponds to the operator $P^t$ parameterized by the diffusion time $t$:

$$D_t(k,l) = \sum_{i=1,\dots,N} \frac{(p_t(k,i) - p_t(l,i))^2}{\pi(i)} \quad \text{where} \quad \pi(i) = \frac{g(i)}{\sum_u g(u)}. \tag{4}$$

The distance $D_t$ is low if there is a large number of paths of length $t$ with high transition probabilities between the nodes $k$ and $l$.

The diffusion distance corresponds to the Euclidean distance in the embedding space: $\|\boldsymbol{\Psi}_t(k) - \boldsymbol{\Psi}_t(l)\| = D_t(k,l)$. The *functional* relations between fMRI signals are translated into spatial distances in the functional geometry [14]. This particular embedding method is closely related to other spectral embedding approaches [17]; the parameter $t$ controls the range of graph nodes that influence a certain local configuration.

To facilitate notation, we assume the diffusion time $t$ is fixed in the remainder of the paper, and omit it from the equations. The resulting maps are the basis for the functional registration of the fMRI volumes.

## 3   Functional geometry alignment

Let $\boldsymbol{\Psi}_0$, and $\boldsymbol{\Psi}_1$ be the functional maps of two subjects. $\boldsymbol{\Psi}_0$, and $\boldsymbol{\Psi}_1$ are point clouds embedded in a $w$-dimensional Euclidean space. The points in the maps correspond to voxels and registration of the maps establishes correspondences between brain regions of the subjects. Our goal is to estimate correspondences of points in the two maps based on the structure in the two distributions determined by the functional connectivity structure in the data. We perform registration in the functional space by non-rigidly deforming the distributions until their overlap is maximized. At the same time, we regularize the deformation so that high frequency displacements of individual points, which would correspond to a change in the relations with strong connectivity, are penalized.

We note that the embedding is defined up to rotation, order and sign of individual coordinate axes. However, for successful alignment it is essential that the embedding is consistent between the subjects, and we have to match the nuisance parameters of the embedding during alignment. In [19] a greedy method for sign matching was proposed. In our data the following procedure produces satisfying results. When computing the embedding, we set the sign of each individual coordinate axis $j$ so that $mean(\{\Psi_j(k)\}) - median(\{\Psi_j(k)\}) > 0, \forall j = 1, \ldots, w$. Since the distributions typically have a long tail, and are centered at the origin, this step disambiguates the coordinate axis directions well.

Fig. 2 illustrates the level of consistency of the maps across two subjects. It shows the first three dimensions of maps for two different control subjects. The colors illustrate clusters in the map and their corresponding positions in the brain. For illustration purposes the colors are matched based on the spatial location of the clusters. The two maps indicate that there is some degree of consistency of the mappings for different subjects.

Eigenvectors may switch if the corresponding eigenvalues are similar [13]. We initialise the registration using Procrustes analysis [4] so that the distance between a randomly chosen subset of vertices from the same anatomical part of the brain is minimised in functional space. This typically resolves ambiguity in the embedding with respect to rotation and the order of eigenvectors in the functional space.

We employ the Coherent Point Drift algorithm for the subsequent non-linear registration of the functional maps [16]. We consider the points in $\mathbf{\Psi}_0$ to be centroids of a Gaussian mixture model that is fitted to the points in $\mathbf{\Psi}_1$ to minimize the energy

$$E(\chi) = -\sum_{k=1}^{N_0} \log \left( \sum_{l=1}^{N_1} exp \left[ -\frac{\|x_0^k - \chi(x_1^l)\|^2}{2\sigma^2} \right] \right) + \frac{\lambda}{2}\phi(\chi), \qquad (5)$$

where $x_0^k$ and $x_1^l$ are the points in the maps $\mathbf{\Psi}_0$ and $\mathbf{\Psi}_1$ during matching and $\phi$ is a function that regularizes the deformation $\chi$ of the point set. The minimization of $E(\chi)$ involves a trade-off between its two terms controlled by $\lambda$. The first term is a Gaussian kernel, that generates a continuous distribution for the entire map $\mathbf{\Psi}_0$ in the functional space $\mathbb{R}^w$. By deforming $x_0^k$ we increase the likelihood of the points in $\mathbf{\Psi}_1$ with respect to the distribution defined by $x_0^k$. At the same time $\phi(\chi)$ encourages a smooth deformation field by penalizing high frequency local deformations by e.g., a radial basis function [15]. The first term in Eq. 5 moves the two point distributions so that their overlap is maximized. That is, regions that exhibit similar global connectivity characteristics are moved closer to each other. The regularization term induces a high penalty on changing strong functional connectivity relationships among voxels (which correspond to small distances or clusters in the map). At the same time, the regularization allows more changes between regions with weak connectivity (which correspond to large distances). In other words, it preserves the connectivity structure of strong networks, while being flexible with respect to weak connectivity between distant clusters.

Once the registration of the two distributions in the functional geometry is completed, we assign correspondences between points in $\mathbf{\Psi}_0$ and $\mathbf{\Psi}_1$ by a simple matching algorithm that for any point in one map chooses the closest point in the other map.

## 4 Validation of Alignment

To validate the functional registration quantitatively we align pairs of subjects via (i) the proposed functional geometry alignment, and (ii) the anatomical non-rigid demons registration [25, 28]. We restrict the functional evaluation to the grey matter. Functional geometry embedding is performed on a random sampling of 8000 points excluding those that exhibit no activation (with a liberal threshold of $p = 0.15$ in the General Linear Model (GLM) analysis [12]). After alignment we evaluate the quality of the fit by (1) the accuracy of predicting the location of the active areas in the target subject and (2) the inter-subject correlation of BOLD signals after alignment. The first criterion is directly related to the clinical aim of localization of active areas.

**A. Predictive power:** We evaluate if it is possible to establish correspondences, so that the activation in one subject lets us predict the activation in another subject after alignment. That is, we examine if the correspondences identify regions that exhibit a relationship with the task (in our experiment, a

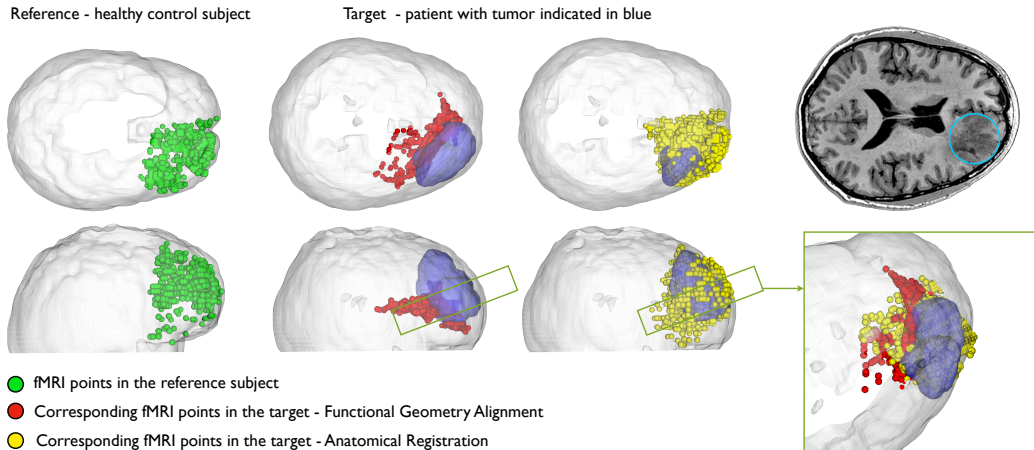

Reference - healthy control subject    Target - patient with tumor indicated in blue

- ● fMRI points in the reference subject
- ● Corresponding fMRI points in the target - Functional Geometry Alignment
- ● Corresponding fMRI points in the target - Anatomical Registration

Figure 3: Mapping a region by functional geometry alignment: a reference subject (first column) aligned to a tumor patient (second and third columns, the tumor is shown in blue). The green region in the healthy subject is mapped to the red region by the proposed functional registration and to the yellow region by anatomical registration. Note that the functional alignment places the region narrowly around the tumor location, while the anatomical registration result intersects with the tumor. The slice of the anatomical scan with the tumor and the zoomed visualization of the registration results (fourth column) are also shown.

language task) even if they are ambiguous or not detected based on the standard single-subject GLM analysis. That is, can we transfer evidence for the activation of specific regions between subjects, for example between healthy controls and tumor patients? In the following we refer to regions detected by the standard single subject fMRI analysis with an activation threshold of $p = 0.05$ (false discovery rate (FDR) corrected [1]) as *above-threshold* regions.

We validate the accuracy of localizing activated regions in a target volume by measuring the average correlation of the t-maps (based on the standard GLM) between the source and the corresponding target regions after registration. A t-map indicates activation - i.e., a significant correlation with the task the subject is performing during fMRI acquisition - for each voxel in the fMRI volume. A high inter-subject correlation of the t-maps indicates that the aligned source t-maps are highly predictive of the t-map in the target fMRI data. Additionally, we measure the overlap between regions in the target image to which the above-threshold source regions are mapped, and the above-threshold regions in the target image. Note that for the registration itself neither the inter-subject correlation of fMRI signals, nor the correlation of t-maps is used. In other words, although we enforce homology in the pattern of correlations between two subjects, the correlations across subjects per se are not matched.

**B. Correlation of BOLD signal across subjects:** To assess the relationship between the source and registered target regions relative to the fMRI activation, we measure the correlation between the fMRI signals in the above-threshold regions of the source volume and the fMRI signals at the corresponding locations in the target volume. Across-subject correlation of the fMRI signals indicates a relationship between the underlying functional processes. We are interested in two specific scenarios: (i) *above*-threshold regions in the target image that were matched to above-threshold regions in the source image, and (ii) *below*-threshold regions in the target image that were matched to above-threshold regions in the source image. This second group includes candidates for activation, even though they do not pass detection threshold in the particular volume. We do not expect correlation of signals for non-activated regions.

## 5 Experimental Results

We demonstrate the method on a set of 6 control subjects and 3 patients with low-grade tumors in one of the regions associated with language processing. For all 9 subjects fMRI data was acquired

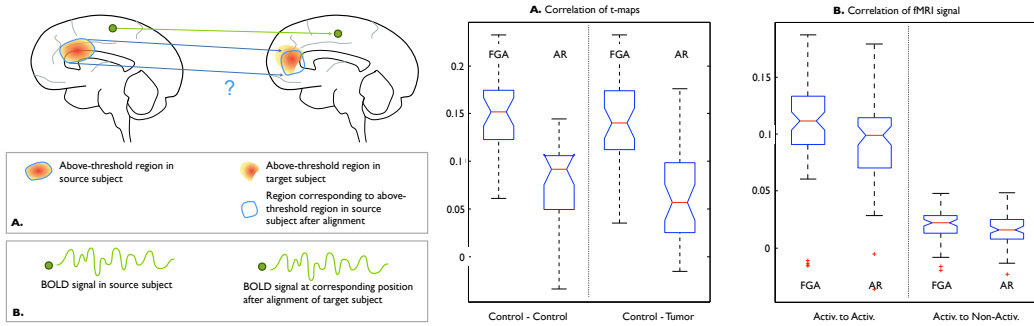

Figure 4: Validation: A. Correlation distribution of corresponding t-values after functional geometry alignment (FGA) and anatomical registration (AR) for control-control and control-tumor matches. B. correlation of the BOLD signals for activated regions mapped to activated regions (left) and activated regions mapped to sub-threshold regions (right).

using a 3T GE Signa system (TR=2s, TE=40ms, flip angle=90°, slice gap=0mm, FOV=25.6cm, dimension $128 \times 128 \times 27$ voxels, voxel size of $2 \times 2 \times 4$ mm$^3$). The language task (antonym generation) block design was 5min 10s, starting with a 10s pre-stimulus period. Eight task and seven rest blocks each 20s long alternated in the design. For each subject, anatomical T1 MRI data was acquired and registered to the functional data. We perform pair-wise registration in all 36 image pairs, 21 of which include at least one patient.

Fig. 3 illustrates the effect of a tumor in a language region, and the corresponding registration results. An area of the brain associated with language is registered from a control subject to a tumor patient. The location of the tumor is shown in blue; the regions resulting from functional and anatomical registration are indicated in red (FGA), and yellow (AR), respectively. While anatomical registration creates a large overlap between the mapped region and the tumor, functional geometry alignment maps the region to a plausible area narrowly surrounding the tumor. Fig. 4 reports quantitative comparison of functional alignment vs. anatomical registration for the entire set of subjects. Functional geometry alignment achieves significantly higher correlation of t-values than anatomical registration ($0.14$ vs. $0.07$, $p < 10^{-17}$, paired t-test, all image pairs). Anatomical registration performance drops significantly when registering a control subject and a tumor patient, compared to a pair of control subjects ($0.08$ vs. $0.06$, $p = 0.007$). For functional geometry alignment this drop is not significant ($0.15$ vs. $0.14$, $p = 0.17$). Functional geometry alignment predicts $50\%$ of the above-threshold in the target brain, while anatomical registration predicts $29\%$ (Fig. 4 (A)).

These findings indicate that the functional alignment of language regions among source and target subjects is less affected by the presence of a tumor and the associated reorganization than the matching of functional regions by anatomical registration. Furthermore the functional alignment has better predictive power for the activated regions in the target subject for both *control-control* and *control-patient* pairs. In our experiments this predictive power is affected onyl to a small degree by a tumor presence in the target. In contrast and as expected, the matching of functional regions by anatomical alignment is affected by the tumor.

Activated source regions mapped to a target subject exhibit the following characteristics. If both source region and corresponding target region are above-threshold the average correlation between the source and target signals is significantly higher for functional geometry alignment ($0.108$ vs. $0.097$, $p = 0.004$ paired t-test). For above-threshold regions mapped to below-threshold regions the same significant difference exists ($0.020$ vs. $0.016$, $p = 0.003$), but correlations are significantly lower. This significant difference between functional geometry alignment and anatomical registration vanishes for regions mapped from below-threshold regions in the source subject. The baseline of below-threshold region pairs exhibits very low correlation ($\sim 0.003$) and no difference between the two methods.

The fMRI signal correlation in the source and the target region is higher for functional alignment if the source region is activated. This suggests that even if the target region does not exhibit task specific behavior detectable by standard analysis, its fMRI signal still correlates with the activated source fMRI signal to a higher degree than non-activated region pairs. The functional connectivity

structure is sufficiently consistent to support an alignment of the functional geometry between subjects. It identifies experimental correspondences between regions, even if their individual relationship to the task is ambiguous. We demonstrate that our alignment improves inter-subject correlation for activated source regions and their target regions, but not for the non-active source regions. This suggest that we enable localization of regions that would not be detected by standard analysis, but whose activations are similar to the source regions in the normal subjects.

# 6   Conclusion

In this paper we propose and demonstrate a method for registering neuroanatomy based on the functional geometry of fMRI signals. The method offers an alternative to anatomical registration; it relies on matching a spectral embedding of the functional connectivity patterns of two fMRI volumes. Initial results indicate that the structure in the diffusion map that reflects functional connectivity enables accurate matching of functional regions. When used to predict the activation in a target fMRI volume the proposed functional registration exhibits higher predictive power than the anatomical registration. Moreover it is more robust to pathologies and the associated changes in the spatial organization of functional areas. The method offers advantages for the localization of activated but displaced regions in cases where tumor-induced changes of the hemodynamics make direct localization difficult. Functional alignment contributes evidence from healthy control subjects. Further research is necessary to evaluate the predictive power of the method for localization of specific functional areas.

**Acknowledgements** This work was funded in part by the NSF IIS/CRCNS 0904625 grant, the NSF CAREER 0642971 grant, the NIH NCRR NAC P41-RR13218, NIH NIBIB NAMIC U54-EB005149, NIH U41RR019703, and NIH P01CA067165 grants, the Brain Science Foundation, and the Klarman Family Foundation.

# References

[1] Y. Benjamini and Y. Hochberg. Controlling the false discovery rate: a practical and powerful approach to multiple testing. *Journal of the Royal Statistical Society. Series B (Methodological)*, pages 289–300, 1995.

[2] S.B. Bonelli, R.H.W. Powell, M. Yogarajah, R.S. Samson, M.R. Symms, P.J. Thompson, M.J. Koepp, and J.S. Duncan. Imaging memory in temporal lobe epilepsy: predicting the effects of temporal lobe resection. *Brain*, 2010.

[3] S. Bookheimer. Pre-surgical language mapping with functional magnetic resonance imaging. *Neuropsychology Review*, 17(2):145–155, 2007.

[4] F.L. Bookstein. Two shape metrics for biomedical outline data: Bending energy, procrustes distance, and the biometrical modeling of shape phenomena. In *Proceedings International Conference on Shape Modeling and Applications*, pages 110 –120, 1997.

[5] Fan R.K. Chung. *Spectral Graph Theory*. American Mathematical Society, 1997.

[6] Ronald R. Coifman and Stéphane Lafon. Diffusion maps. *App. Comp. Harm. An.*, 21:5–30, 2006.

[7] Bryan Conroy, Ben Singer, James Haxby, and Peter Ramadge. fmri-based inter-subject cortical alignment using functional connectivity. In *Adv. in Neural Information Proc. Systems*, pages 378–386, 2009.

[8] E. Fedorenko and N. Kanwisher. Neuroimaging of Language: Why Hasn't a Clearer Picture Emerged? *Language and Linguistics Compass*, 3(4):839–865, 2009.

[9] B. Fischl, M.I. Sereno, and A.M. Dale. Cortical surface-based analysis II: Inflation, flattening, and a surface-based coordinate system. *Neuroimage*, 9(2):195–207, 1999.

[10] B. Fischl, M.I. Sereno, R.B.H. Tootell, and A.M. Dale. High-resolution intersubject averaging and a coordinate system for the cortical surface. *HBM*, 8(4):272–284, 1999.

[11] KJ Friston, CD Frith, P. Fletcher, PF Liddle, and RSJ Frackowiak. Functional topography: multidimensional scaling and functional connectivity in the brain. *Cerebral Cortex*, 6(2):156, 1996.

[12] KJ Friston, AP Holmes, KJ Worsley, JB Poline, CD Frith, RSJ Frackowiak, et al. Statistical parametric maps in functional imaging: a general linear approach. *Hum Brain Mapp*, 2(4):189–210, 1995.

[13] V. Jain and H. Zhang. Robust 3D shape correspondence in the spectral domain. In *Shape Modeling and Applications, 2006. SMI 2006. IEEE International Conference on*, page 19. IEEE, 2006.

[14] Georg Langs, Dimitris Samaras, Nikos Paragios, Jean Honorio, Nelly Alia-Klein, Dardo Tomasi, Nora D Volkow, and Rita Z Goldstein. Task-specific functional brain geometry from model maps. In *Proc. of MICCAI*, volume 11, pages 925–933, 2008.

[15] A. Myronenko and X. Song. Point set registration: Coherent point drift. *IEEE Transactions on Pattern Analysis and Machine Intelligence*, 2010.

[16] A. Myronenko, X. Song, and M.A. Carreira-Perpinán. Non-rigid point set registration: Coherent Point Drift. *Adv. in Neural Information Proc. Systems*, 19:1009, 2007.

[17] H.J. Qiu and E.R. Hancock. Clustering and Embedding Using Commute Times. *IEEE TPAMI*, 29(11):1873–1890, 2007.

[18] M.R. Sabuncu, B.D. Singer, B. Conroy, R.E. Bryan, P.J. Ramadge, and J.V. Haxby. Function-based intersubject alignment of human cortical anatomy. *Cerebral Cortex*, 20(1):130–140, 2010.

[19] L.S. Shapiro and J. Michael Brady. Feature-based correspondence: an eigenvector approach. *Image and vision computing*, 10(5):283–288, 1992.

[20] D. Shen and C. Davatzikos. HAMMER: hierarchical attribute matching mechanism for elastic registration. *IEEE Trans. Med. Imaging*, 21(11):1421–1439, 2002.

[21] J. Talairach and P. Tournoux. *Co-planar stereotaxic atlas of the human brain*. Thieme New York, 1988.

[22] B. Thirion, G. Flandin, P. Pinel, A. Roche, P. Ciuciu, and J.B. Poline. Dealing with the shortcomings of spatial normalization: Multi-subject parcellation of fMRI datasets. *Human brain mapping*, 27(8):678–693, 2006.

[23] B. Thirion, P. Pinel, S. Mériaux, A. Roche, S. Dehaene, and J.B. Poline. Analysis of a large fMRI cohort: Statistical and methodological issues for group analyses. *Neuroimage*, 35(1):105–120, 2007.

[24] Bertrand Thirion, Silke Dodel, and Jean-Baptiste Poline. Detection of signal synchronizations in resting-state fmri datasets. *Neuroimage*, 29(1):321–327, 2006.

[25] J.P. Thirion. Image matching as a diffusion process: an analogy with Maxwell's demons. *Medical Image Analysis*, 2(3):243–260, 1998.

[26] D.C. Van Essen, H.A. Drury, J. Dickson, J. Harwell, D. Hanlon, and C.H. Anderson. An integrated software suite for surface-based analyses of cerebral cortex. *Journal of the American Medical Informatics Association*, 8(5):443, 2001.

[27] U. Von Luxburg. A tutorial on spectral clustering. *Statistics and Computing*, 17(4):395–416, 2007.

[28] H. Wang, L. Dong, J. O'Daniel, R. Mohan, A.S. Garden, K.K. Ang, D.A. Kuban, M. Bonnen, J.Y. Chang, and R. Cheung. Validation of an accelerated'demons' algorithm for deformable image registration in radiation therapy. *Physics in Medicine and Biology*, 50(12):2887–2906, 2005.

